# Analyzing 3D Objects in Cluttered Images

**Mohsen Hejrati**
UC Irvine
shejrati@ics.uci.edu

**Deva Ramanan**
UC Irvine
dramanan@ics.uci.edu

## Abstract

We present an approach to detecting and analyzing the 3D configuration of objects in real-world images with heavy occlusion and clutter. We focus on the application of finding and analyzing cars. We do so with a two-stage model; the first stage reasons about 2D shape and appearance variation due to within-class variation (station wagons look different than sedans) and changes in viewpoint. Rather than using a view-based model, we describe a compositional representation that models a large number of effective views and shapes using a small number of local view-based templates. We use this model to propose candidate detections and 2D estimates of shape. These estimates are then refined by our second stage, using an explicit 3D model of shape and viewpoint. We use a morphable model to capture 3D within-class variation, and use a weak-perspective camera model to capture viewpoint. We learn all model parameters from 2D annotations. We demonstrate state-of-the-art accuracy for detection, viewpoint estimation, and 3D shape reconstruction on challenging images from the PASCAL VOC 2011 dataset.

## 1  Introduction

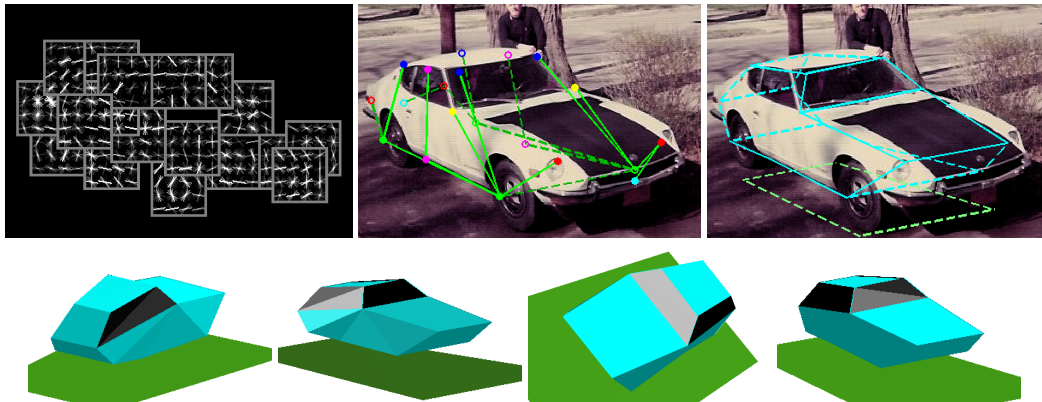

Figure 1: We describe two-stage models for detecting and analyzing the 3D shape of objects in unconstrained images. In the first stage, our models reason about 2D appearance and shape using variants of deformable part models (DPMs). We use global mixtures of trees with local mixtures of gradient-based part templates (**top-left**). Global mixtures capture constraints on visibility and shape (headlights are only visible in certain views at certain locations), while local mixtures capture constraints on appearance (headlights look different in different views). Our 2D models localize even fully-occluded landmarks, shown as hollow circles and dashed lines in (**top-middle**). We feed this output to our second stage, which directly reasons about 3D shape and camera viewpoint. We show the reconstructed 3D model and associated ground-plane (assuming its parallel to the car body) on (**top-right**). The **bottom** row shows 3D reconstructions from four novel viewpoints.

A grand challenge in machine vision is the task of understanding 3D objects from 2D images. Classic approaches based on 3D geometric models [2] could sometimes exhibit brittle behavior on cluttered, "in-the-wild" images. Contemporary recognition methods tend to build statistical models of 2D appearance, consisting of classifiers trained with large training sets using engineered appearance features. Successful examples include face detectors [30], pedestrian detectors [7], and general

object-category detectors [10]. Such methods seem to work well even in cluttered scenes, but are usually limited to coarse 2D output, such as bounding-boxes.

Our work is an attempt to combine the two approaches, with a focus on statistical, 3D geometric models of objects. Specifically, we focus on the practical application of detecting and analyzing cars in cluttered, unconstrained images. We refer the reader to our results (Fig.4) for a sampling of cluttered images that we consider. We develop a model that detects cars, estimates camera viewpoint, and recovers 3D landmarks configurations and their visibility with state-of-the-art accuracy. It does so by reasoning about appearance, 3D shape, and camera viewpoint through the use of 2D structured, relational classifiers and 3D geometric subspace models.

While deformable models and pictorial structures [10, 31, 11] are known to successfully model articulation, 3D viewpoint is still not well understood. The typical solution is to "discretize" viewpoint and build multiple view-based models tuned for each view (frontal, side, 3/4...). One advantage of such a "brute-force" approach is that it is computationally efficient, at least for a small number of views. Fine-grained 3D shape estimation may still be difficult with such a strategy. On the other hand, it is difficult to build models that reason directly in 3D because the "inverse-rendering" problem is hard to solve. We introduce a two-stage approach that first reasons about 2D shape and appearance variation, and then reasons explicitly about 3D shape and viewpoint given 2D correspondences from the first stage. We show that "inverse-rendering" *is* feasible by way of 2D correspondences.

**2D shape and appearance:** Our first stage models 2D shape and appearance using a variant of deformable part models (DPMs) designed to produce reliable 2D landmark correspondences. Our approach differs from traditional view-based models in that it is *compositional*; it "cuts and pastes" together different sets of local view-based templates to model a large set of global viewpoints. We use global mixtures of trees with local mixtures of "part" or landmark templates. Global mixtures capture constraints on visibility and shape (headlights are only visible in certain views at certain locations), while local mixtures capture constraints on appearance (headlights look different in different views). We use this model to efficiently generate candidate 2D detections that are refined by our second 3D stage. One salient aspect of our 2D model is that it reports 2D locations of all landmarks including occluded ones, each augmented with a visibility flag.

**3D shape and viewpoint:** Our second layer processes the 2D output of our first stage, incorporating global shape constraints arising from 3D shape variation and viewpoint. To capture viewpoint constraints, we model landmarks as weak-perspective projections of a 3D object. To capture within-class variation, we model the 3D shape of any object instance as a linear combination of 3D basis shapes. We use tools from nonrigid structure-from-motion (SFM) to both learn and enforce such models using 2D correspondences. Crucially, we make use of occlusion reports generated by our local view-based templates to estimate morphable 3D shape and camera viewpoint.

## 2 Related Work

We focus most on recognition methods that deal explicitly with 3D viewpoint variation.

**Voting-based methods:** One approach to detection and viewpoint classification is based on bottom-up geometric voting, using a Hough transform or geometric hashing. Images are first processed to obtain a set of local feature detections. Each detection can then vote for both an object location and viewpoint. Examples include [12] and implicit shape models [1, 26]. Our approach differs in that we require no initial feature detection stage, and instead we reason about all possible geometric configurations and occlusion states.

**View-based models:** Early successful approaches included multiview face detection [24, 17]. Recent approaches based on view-based deformable part models include [19, 13, 10]. Our model differs in that we use a single representation that directly generates multiple views. One can augment view-based models to share local parts across views [27, 21, 32]. This typically requires reasoning about topological changes in viewpoint; certain parts or features can only be visible in certain view due to self-occlusion. One classic representation for encoding such visibility constraints is an aspect graph [5]. [33] model such topological constraints with global mixtures with varying tree structures. Our model is similar to such approaches, except that we use a decomposable notion of aspect; we simultaneously reason about global and semi-local changes in visibility using local part mixtures with global co-occurrence constraints.

**3D models:** One can also directly reason about local features and their geometric arrangement in a 3D coordinate system [23, 25, 34]. Though such models are three-dimensional in terms of their underlying representation, run-time inference usually proceeds in a bottom-up manner, where detected features vote for object locations. To handle non-Gaussian observation models, [18] evaluate randomly sampled model estimates within a RANSAC search. Our approach is closely related to the recent work of [22], which also uses a deformable part model (DPM) to capture viewpoint variation in cars. Though they learn spatial constraints in a 3D coordinate frame, their model at run-time is equivalent to a view-based model, where each view is modeled with a star-structured DPM. Our model differs in that we directly reason about the location of fully-occluded landmarks, we model an exponential number of viewpoints by using a compositional representation, and we produce continuous 3D shapes and camera viewpoints associated with each detection using only 2D training data. Finally, we represent the space of 3D models of an object category using a set of basis shapes, similar to the morphable models of [3]. To estimate such models from 2D data, we adapt methods designed for tracking morphable shapes to 3D object category recognition [29, 28].

## 3 2D Shape and Appearance

We first describe our 2D model of shape and appearance. We write it as a scoring function with linear parameters. Our model can be seen as an extension of the flexible mixtures-of-part model [31], which itself augments a deformable part model (DPM) [10] to reason about local mixtures. Our model differs its encoding of occlusion states using local mixtures, as well as the introduction of global mixtures that enforce occlusions and spatial geometry consistent with changes in 3D viewpoint. We take care to design our model so as to allow for efficient dynamic-programming algorithms for inference.

Let $I$ be an image, $p_i = (x, y)$ be the pixel location for part $i$ and $t_i \in \{1..T\}$ be the local mixture component of part $i$. As an example, part $i$ may correspond to a front-left headlight, and $t_i$ can correspond to different appearances of a headlight in frontal, side, or three-quarter views. A notable aspect of our model is that we estimate landmark locations for all parts in all views, even when they are fully occluded. We will show that local mixture variables perform surprisingly well at modeling complex appearances arising from occlusions.

Let $i \in V$ where $V$ is the set of all landmarks. We consider different relational graphs $G_m = (V, E_m)$ where $E_m$ connects pairs of landmarks constrained to have consistent locations and local mixtures in global mixture $m$. We can loosely think of $m$ as a "global viewpoint", though it will be latently estimated from the data. We use the lack of subscript to denote the set of variables obtained by iterating over that subscript; e.g., $p = \{p_i : i \in V\}$. Given an image, we score a collection of landmark locations and mixture variables

$$S(I, p, t, m) = \sum_{i \in V} \left[ \alpha_i^{t_i} \cdot \phi(I, p_i) \right] + \sum_{ij \in E_m} \left[ \beta_{ijm}^{t_i, t_j} \cdot \psi(p_i - p_j) + \gamma_{ijm}^{t_i, t_j} \right] \quad (1)$$

**Local model:** The first term scores the appearance evidence for placing a template $\alpha_i^{t_i}$ for part $i$, tuned for mixture $t_i$, at location $p_i$. We write $\phi(I, p_i)$ for the feature vector (e.g., HOG descriptor [7]) extracted from pixel location $p_i$ in image $I$. Note that we define a template even for mixtures $t_i$ corresponding to fully-occluded states. One may argue that no image evidence should be scored during an occlusion; we take the view that the learning algorithm can decide for itself. It may choose to learn a template of all zeros (essentially ignoring image evidence) or it may find gradient features statistically correlated with occlusions (such as t-junctions). Unlike the remaining terms in our scoring function, the local appearance model is *not* dependent on the global mixture/viewpoint. We show that this independence allows our model to compose together different local mixtures to model a single global viewpoint.

**Relational model:** The second term scores relational constraints between pairs of parts. We write $\psi(p_i - p_j) = \begin{bmatrix} dx & dx^2 & dy & dy^2 \end{bmatrix}$, a vector of relative offsets between part $i$ and part $j$. We can interpret $\beta_{ijm}^{t_i, t_j}$ as the parameters of a spring specifying the relative rest location and quadratic spring penalty for deviating from that rest location. Notably, this spring depends on part $i$ and $j$, the local mixture components of part $i$ and $j$, and the global mixture $m$. This dependency captures many natural constraints due to self-occlusion; for example, if a car's left-front wheel lies to the right of the other front wheel (in image space), than it is likely self-occluded. Hence it is crucial that local appearance and geometry depend on each other. The last term $\gamma_{ijm}^{t_i, t_j}$ defines a co-occurrence score associated with instancing local mixture $t_i$ and $t_j$, and global mixture $m$. This encodes the

constraint that, if the left front headlight is occluded due to self occlusion, the left front wheel is also likely occluded.

**Global model:** We define different graphs $G_m = (V, E_m)$ corresponding to different global mixtures. We can loosely think of the global variable $m$ are capturing a coarse, quantized viewpoint. To ensure tractability, we force all edge structures to be tree-structured. Intuitively, different relational structures may help because occluded landmarks tend to be localized with less reliability. One may expect occluded/unreliable parts should have fewer connections (lower degrees in $G_m$) than reliable parts. Even for a fixed global mixture $m$, our model can generate an exponentially-large set of appearances $|V|^T$, where $T$ is the number of local mixture types. We show such a model outperforms a naive view-based model in our experiments.

## 3.1 Inference

Inference corresponds to maximizing (1) with respect to landmark locations $p$, local mixtures $t$, and global mixtures $m$:

$$S^*(I) = \max_m [\max_{p,t} S(I, p, t, m)] \tag{2}$$

We optimize the above equation by enumerating all global mixtures $m$, and for each global mixture, finding the optimal combination of landmark locations $p$ and local mixtures $t$ by dynamic programing (DP). To see that the inner maximization can be optimized by DP, let us define $z_i = (p_i, t_i)$ to denote both the discrete pixel position and discrete mixture type of part $i$. We can rewrite the score from (1) for a fixed image $I$ and global mixture $m$ with edge structure $E$ as:

$$S(z) = \sum_{i \in V} \phi_i(z_i) + \sum_{ij \in E} \psi_{ij}(z_i, z_j), \quad \text{(for a fixed } I \text{ and } m) \tag{3}$$

$$\text{where} \quad \phi_i(z_i) = \alpha_i^{t_i} \cdot \phi(I, p_i) \quad \text{and} \quad \psi_{ij}(z_i, z_j) = \beta_{ijm}^{t_i, t_j} \cdot \psi(p_i - p_j) + \gamma_{ijm}^{t_i, t_j}$$

From this perspective, it is clear that our model (conditioned on $I$ and $m$) is a discrete, pairwise Markov random field (MRF). When $G = (V, E)$ is tree-structured, one can compute $\max_z S(z)$ with dynamic programming [31].

## 3.2 Learning

We assume we are given training data consisting of image-landmark triplet $\{I_n, p_{in}, o_{in}\}$, where landmarks are augmented with an additional discrete visibility flag $o_{in}$. With a slight abuse of notation, we use $n$ to denote an instance of a training image. We use $o_{in} \in \{0, 1, 2\}$ to denote visible, self-occlusion, and other-occlusion respectively, where other occlusion corresponds to a landmark that is occluded by another object (or the image border). We now show how to augment this training set with local mixtures labels $t_{in}$, global mixtures labels $m_n$, and global edge structures $E_m$. Essentially, we infer such mixture labels using probabilistic algorithms for generating local/global clusters of 2D landmark configurations. We then use this inferred mixture labels to train the linear parameters of the scoring function (1) using supervised, max-margin methods.

**Learning local mixtures:** We use the clustering algorithm described in [8, 4] to learn local part mixtures. We construct a "local-geometric-context" vector for each part, and obtain landmark mixture labels by grouping landmark instances with similar local geometry. Specifically, for each landmark $i$ and image $n$, we construct a $K$-element vector $g_{in}$ that defines the 2D relative location of a landmark with respect to the other $K$ landmarks in instance $n$, normalized for the size of that training instance. We construct sets of features $\text{Set}_{ij} = \{g_{in} : n \in 1..N \text{ and } o_{in} = j\}$ corresponding to each part $i$ and occlusion state $j$. We separately cluster each set of vectors using $K$-means, and then interpret cluster membership as mixture label $t_{in}$. This means that, for landmark $i$, a third of its $T$ local mixtures will model visible instances in the training set, a third will model self-occlusions, and a third will capture other-occlusions.

**Learning relational structure:** Given local mixture labels $t_{in}$, we simultaneously learn global mixtures $m_n$ and edge structure $E_m$ with a probabilistic model of $z_{in} = (p_{in}, t_{in})$. We find the global mixtures and edge structure that maximizes the probability of the observed $\{z_{in}\}$ labels. Probabilistically speaking, our spatial spring model is equivalent to a Gaussian model (who's mean and covariance correspond to the rest location and rigidity), making estimation relatively straightforward. We first describe the special case of a single global mixture, for which the most-likely tree $E$ can be obtained by maximizing the mutual information of the labels using the Chow-Liu algorithm

[6, 15]. In our case, we find the maximum-weight spanning tree in a fully connected graph whose edges are labeled with the mutual information (MI) between $z_i = (p_i, t_i)$ and $z_j = (p_j, t_j)$:

$$MI(z_i, z_j) = MI(t_i, t_j) + \sum_{t_i, t_j} P(t_i, t_j) MI(p_i, p_j | t_i, t_j) \qquad (4)$$

$MI(t_i, t_j)$ can be directly computed from the empirical joint frequency of mixture labels in the training set. $MI(p_i, p_j | t_i, t_j)$ is the mutual information of the Gaussian random variables for the location of landmarks $i$ and $j$ given a fixed pair of discrete mixture types $t_i, t_j$; this again is readily obtained by computing the determinant of the sample covariance of the locations of landmarks $i$ and $j$, estimated from the training data. Hence both spatial consistency and mixture consistency are used when learning our relational structure.

**Learning structure and global mixtures:** To simultaneously learn global mixture labels $m_n$ and edge structures associated with each mixture $E_m$, we use an EM algorithm for learning mixtures of trees [20, 15]. Specifically, Meila and Jordan [20] describe an EM algorithm that iterates between inferring distributions over tree mixture assignments (the E-step) and estimating the tree structure (the M-step). One can write the expected complete log-likelihood of the observed labels $\{z\}$, where $\theta$ are the model parameters (Gaussian spatial models, local mixture co-occurrences and global mixture priors) to be maximized and the global mixture assignment variables $\{m_n\}$ are the hidden variables to be marginalized. Notably, the M-step makes use of the Chow-Liu algorithm. We omit detailed equations for lack of space, but note that this is a relatively straightforward application of [20]. We demonstrate that our latently-estimated global mixtures are crucial for high-performance in 3D reasoning.

**Learning parameters:** The previous steps produces local/global mixture labels and edge structures. Treating these as "ground-truth", we now define a supervised max-margin framework for learning model parameters. To do so, let us write the landmark position labels $p_n$, local mixtures labels $t_n$, and global mixture label $m_n$ collectively as $y_n$. Given a training set of positive images with labels $\{I_n, y_n\}$ and negative images not containing the object of interest, we define a structured prediction objective function similar to one proposed in [31]. The scoring function in (1) is linear in the parameters $w = \{\alpha, \beta, \gamma\}$, and therefore can be expressed as $S(I_n, y_n) = w \cdot \Phi(I_n, y_n)$. We learn a model of the form:

$$\underset{w, \xi_i \geq 0}{\operatorname{argmin}} \quad \frac{1}{2} w^T \cdot w + C \sum_n \xi_n \qquad (5)$$

$$\text{s.t.} \quad \forall n \in \text{positive images} \quad w \cdot \Phi(I_n, y_n) \geq 1 - \xi_n$$
$$\forall n \in \text{negative images}, \forall y \quad w \cdot \Phi(I_n, y) \leq -1 + \xi_n$$

The above constraint states that positive examples should score better than 1 (the margin), while negative examples, for all configurations of part positions and mixtures, should score less than -1. We collect negative examples from images that does not contain any cars. This form of learning problem is known as a structural SVM, and there exist many well-tuned solvers such as the cutting plane solver of SVMStruct in [16] and the stochastic gradient descent solver in [10]. We use the dual coordinate-descent QP solver of [31]. We show an example of a learned model and its learned tree structure in Fig.1.

## 4   3D Shape and Viewpoint

The previous section describes our 2D model of appearance and shape. We use it to propose detections with associated landmarks positions $p^*$. In this section, we describe a 3D shape and viewpoint model for refining $p^*$. Consider 2D views of a single rigid object; 2D landmark positions must obey epipolar geometry constraints. In our case, we must account for within-class shape variation as well (e.g., sedans look different than station wagons). To do so, we make two simplifying assumptions: (1) We assume depth variation of our objects are small compared to the distance from the camera, which corresponds to a weak-perspective camera model. (2) We assume the 3D landmarks of all object instances can be written as linear combinations of a few basis shapes. Let us write the set of detected landmark positions as $p^*$ as a $2 \times K$ matrix where $K = |V|$. We now describe a procedure for refining $p^*$ to be consistent with these two assumptions:

$$\min_{R, \alpha} ||p^* - R \sum_i \alpha_i B_i||^2 \quad \text{where} \quad p \in \mathbb{R}^{2 \times K}, R \in \mathbb{R}^{2 \times 3}, RR^T = Id, B_i \in \mathbb{R}^{3 \times K} \qquad (6)$$

Here, $R$ is an orthonormal camera projection matrix and $B_i$ is the $i^{th}$ basis shape, and $Id$ is the identity matrix. We factor out camera translations by working with mean-centered points $p^*$ and let $\alpha$ directly model weak-perspective scalings.

**Inference:** Given 2D landmark locations $p^*$ and a known set of 3D basis shapes $B^i$, inference corresponds to minimizing (6). For a single basis shape ($n_B = 1$), this problem is equivalent to the well-known "extrinsic orientation" problem of registering a 3D point cloud to a 2D point cloud with known correspondence [14]. Because the squared error is linear in $a_i$ and $R$, we solve for the coefficients and rotation with an iterative least-squares algorithm. We enforce the orthonormality of $R$ with a nonlinear optimization, initialized by the least-squares solution [14]. This means that we can associate each detection with shape basis coefficients $\alpha_i$ (which allows us to reconstruct the 3D shape) and camera viewpoint $R$. One could combine the reprojection error of (6) with our original scoring function from (1) into a single objective that jointly searches over all 2D and 3D unknowns. However inference would be exponential in $K$. We find a two-layer inference algorithm to be computationally efficient but still effective.

**Learning:** The above inference algorithm requires the morphable 3D basis $B_i$ at test-time. One can estimate such a basis given training data with labeled 2D landmark positions by casting this as nonrigid structure from motion (SFM) problem. Stack all 2D landmarks from $N$ training images into a $2N \times K$ matrix. In the noise-free case, this matrix is rank $3n_B$ (where $n_B$ is the number of basis shapes), since each row can be written as a linear combination of the 3D coordinates of $n_B$ basis shapes. This means that one can use rank constraints to learn a 3D morphable basis. We use the publically-available nonrigid SFM code [28]. By slightly modifying it to estimate "motion" given a known "structure", we can also use it to perform the previous projection step during inference.

**Occlusion:** A well-known limitation of SFM methods is their restricted success under heavy occlusion. Notably, our 2D appearance model provides location estimates for occluded landmarks. Many SFM methods (including [28]) can deal with limited occlusion through the use of low-rank constraints; essentially, one can still estimate low-rank approximations of matrices with some missing entries. We can use this property to learn models from partially-labeled training sets. Recall that our learning formulation requires all landmarks (including occluded ones) to be labeled in training data. Manually labeling the positions of occluded landmarks can be ambiguous. Instead, we use the estimated shape basis and camera viewpoints to infer/correct the locations of occluded landmarks.

## 5  Experiments

**Datasets:** To evaluate our model, we focus on car detection and 3D landmark estimation in cluttered, real-world datasets with severe occlusions. We labeled a subset of 500 images from the PASCAL VOC 2011 dataset [9] with locations and visibility states of 20 car landmarks. Our dataset contains 723 car instances. 36% of landmarks are not visible due to self-occlusion, while 21% of landmarks are not visible due to occlusion by another object (or truncation due to the image border). Hence *over half* our landmarks are occluded, making our dataset considerably more difficult than those typically used for landmark localization or 3D viewpoint estimation. We evenly split the images into a train/test set. We also compare results on a more standard viewpoint dataset from [1], which consists of 200 relatively "clean" cars from the PASCAL VOC 2007 dataset, marked with 40 discrete viewpoint class labels.

**Implementation:** We modify the publically-available code of [31] and [28] to learn our models, setting the number of local mixtures $T = 9$, the number of global mixtures $M = 50$, and the number of basis shapes $n_B = 5$. We found results relatively robust to these settings. Learning our 2D deformable model takes roughly 4 hours, while learning our 3D shape model takes less than a minute. Our model is defined at a canonical scale, so we search over an image pyramid to find detections at multiple scales. Total run-time for a test image (including both 2D and 3D processing over all scales) is 10 seconds.

**Evaluation:** Given an image, our algorithm produces multiple detections, each with 3D landmark locations, visibility flags, and camera viewpoints. We qualitatively visualize such output in Fig.4. To evaluate our output, we assume test images are marked with ground-truth cars, each annotated with ground-truth 2D landmarks and visibility flags. We measure the performance of our algorithm on four tasks. We evaluate **object detection (AP)** using using the PASCAL criteria of Average Precision [9], defining a detection to be correct if its bounding box overlaps the ground truth by 50% or more. We evaluate **2D landmark localization (LP)** by counting the fraction of predicted

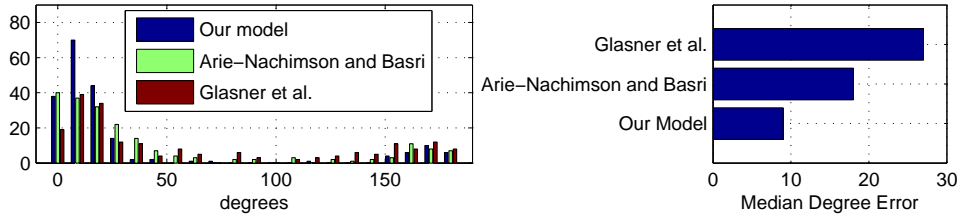

Figure 2: We report histograms of viewpoint label errors for the dataset of [1]. We compare to the reported performance of [1] and [12]. Our model reduces the median error (**right**) by a factor of 2.

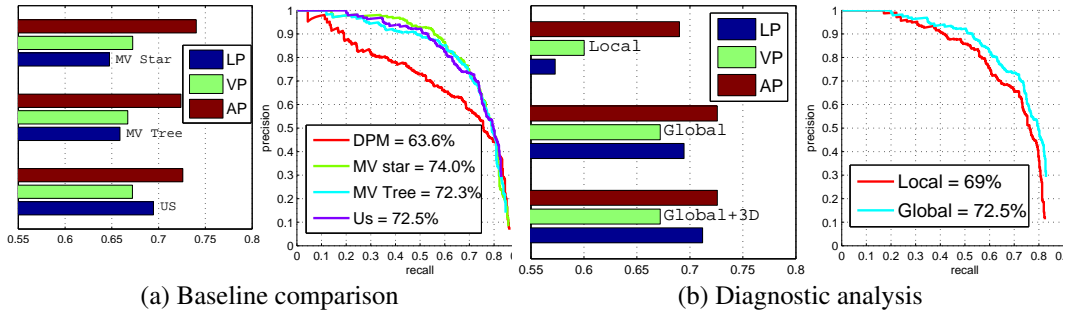

(a) Baseline comparison     (b) Diagnostic analysis

Figure 3: We compare our model with various view-based baselines in (a), and examine various components of our model through a diagnostic analysis in (b). We refer the reader to the text for a detailed analysis, but our model outperforms many state-of-the-art view-based baselines based on trees, stars, and latent parts. We also find that modeling the effects of shape due to global changes in 3D viewpoint is crucial for both detection and landmark localization.

landmarks that lie within $.5x$ pixels of the ground-truth, where $x$ is the diameter of the associated ground-truth wheel. We evaluate **landmark visibility prediction (VP)** by counting the number of landmarks whose predicted visibility state matches the ground-truth, where landmarks may be "visible", "self-occluded", or "other-occluded". Our 3D shape model refines only LP and VP, so AP is determined solely by our 2D (mixtures of trees) model. To avoid conflating the evaluation measures, we evaluate LP and VP assuming bounding-box correspondences between candidates and ground-truth instances are provided. Finally to evaluate **viewpoint classification (VC)**, we compare predicted camera viewpoints with ground-truth viewpoints on the standard benchmark of [1].

**Viewpoint Classification:** We first present results for viewpoint classification in Fig.2 on the benchmark of [1]. Given a test instance, we run our detector, estimate the camera rotation $R$, and report the reconstructed 2D landmarks generated using the estimated $R$. Then we produce a quantized viewpoint label by matching the reconstructions to landmark locations for a reference image (provided in the dataset). We found this approach more reliable than directly matching 3D rotation matrices (for which metric distances are hard to define). We produce a median error of 9 degrees, a factor of 2 improvement over state-of-the-art. This suggests our model does accurately capture viewpoints. We next turn to a detailed analysis on our new cluttered dataset.

**Baselines:** We compare the performance of our overall system to several existing approaches for multiview detection in Fig.3(a). We first compare to widely-used latent deformable part model (DPM) of [10], trained on the exact same data as our model. A supervised DPM (MV-star) considerably improves performance from 63 to 74% AP, where supervision is provided for (view-specific) root mixtures and part locations. This latter model is equivalent in structure to a state-of-the-art model for car detection and viewpoint estimation [22], which trains a DPM using supervision provided by a 3D CAD model. By allowing for tree-structured relations in each view-specific global mixture (MV-tree), we see a small drop in AP = 72.3%. Our final model is similar in term of detection performance (AP = 72.5%), but does noticeably better than both view-based models for landmark prediction. We correctly localize landmarks 69.5% of time, while MV-tree and MV-star score 65.7% and 64.7%, respectively. We produce landmark visibility (VP) estimates from our multiview baselines by predicting a fixed set of visibility labels conditioned on the view-based mixture. We should note that accurate landmark localization is crucial for estimating the 3D shape of the detected instance. We attribute our improvement to the fact that our model can model a large number of global viewpoints by composing together different local view-based templates.

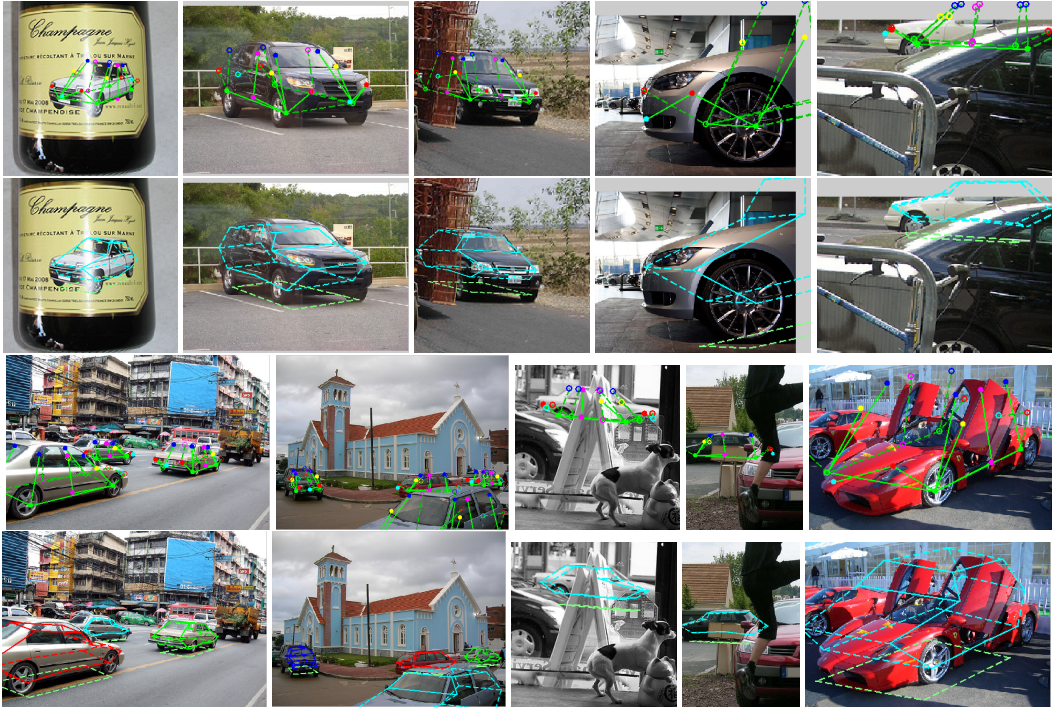

Figure 4: Sample results of our system on real images with heavy clutter and occlusion. We show pairs of images corresponding to detections that matched to ground-truth annotations. The top image (in the pair) shows the output of our tree model, and the bottom shows our 3D shape reconstruction, following the notational conventions of Fig.1. Our system estimates 3D shapes of multiple cars under heavy clutter and occlusions, even in cases where more than 50% of a car is occluded. Our morphable 3D model adapts to the shape of the car, producing different reconstructions for SUVs and sedans (row 2, columns 2-3). Recall that our tree model explicitly reasons about changes in visibility due to self-occlusions versus occlusions from other objects, manifested as local mixture templates. This allow our 3D reconstructions to model occlusions due to other objects (e.g., the rear of the car in row 2, column 3). In some cases, the estimated 3D shape is misaligned due to extreme shape variation of the car instance (e.g., the folding doors on the lower-right).

**Diagnostics:** We compare various aspects of our model in Fig.3(b). "Local" refers to a single tree model with local mixtures only, while "Global" refers to our global mixtures of trees. We see a small improvement in terms of AP, from 69% for "Local" to 72.5% for "Global". However, in terms of landmark prediction, "Global" strongly outperforms "Local", 69.4% to 57.2%. We use these predicted landmarks to estimate 3D shape below.

**3D Shape:** Our 3D shape model reports back a $z$ depth value for each landmark $(x, y)$ position. Unfortunately, depth is hard to evaluate without ground-truth 3D annotations. Instead, we evaluate the improvement in re-projected VP and LP due to our 3D shape model; we see a small 2% improvement in LP accuracy, from 69.4% to 71.2%. We further analyze this by looking at the improvement in localization accuracy of ground-truth landmarks that are visible (73.3 to 74.8), self-occluded (70.5 to 72.5), and other-occluded (22.5 to 23.4%). We see the largest improvement for occluded parts, which makes intuitive sense. Local templates corresponding to occluded mixtures will be less accurate, and so will benefit more from a 3D shape model.

**Conclusion:** We have described a geometric model for detecting and estimating the 3D shape of objects in heavily cluttered, occluded, real-world images. Our model differs from typical multiview approaches by reasoning about local changes in landmark appearance and global changes in visibility and shape, through the aid of a morphable 3D model. While our model is similar to prior work in terms of detection performance, it produces significantly better estimates of 2D/3D landmarks and camera positions, and quantifiably improves localization of occluded landmarks. Though we have focused on the application of analyzing cars, we believe our method could apply to other geometrically-constrained objects.

# References

[1] M. Arie-Nachimson and R. Basri. Constructing implicit 3d shape models for pose estimation. In *ICCV*, 2009.

[2] T. Binford. Survey of model-based image analysis systems. *The International Journal of Robotics Research*, 1(1):18–64, 1982.

[3] V. Blanz and T. Vetter. A morphable model for the synthesis of 3d faces. In *Proceedings of the 26th annual conference on Computer graphics and interactive techniques*, pages 187–194. ACM Press/Addison-Wesley Publishing Co., 1999.

[4] L. Bourdev and J. Malik. Poselets: Body part detectors trained using 3d human pose annotations. In *Computer Vision, 2009 IEEE 12th International Conference on*, pages 1365–1372. IEEE, 2009.

[5] K. Bowyer and C. Dyer. Aspect graphs: An introduction and survey of recent results. *International Journal of Imaging Systems and Technology*, 2(4):315–328, 1990.

[6] C. Chow and C. Liu. Approximating discrete probability distributions with dependence trees. *Information Theory, IEEE Transactions on*, 14(3):462–467, 1968.

[7] N. Dalal and B. Triggs. Histograms of oriented gradients for human detection. In *CVPR*, 2005.

[8] C. Desai and D. Ramanan. Detecting actions, poses, and objects with relational phraselets. *ECCV*, 2012.

[9] M. Everingham, L. Van Gool, C. K. I. Williams, J. Winn, and A. Zisserman. The PASCAL Visual Object Classes Challenge 2011 (VOC2011) Results. http://www.pascal-network.org/challenges/VOC/voc2011/workshop/index.html.

[10] P. F. Felzenszwalb, R. B. Girshick, D. McAllester, and D. Ramanan. Object detection with discriminatively trained part based models. *IEEE PAMI*, 99(1), 5555.

[11] R. Girshick, P. Felzenszwalb, and D. McAllester. Object detection with grammar models. In *NIPS*, 2011.

[12] D. Glasner, M. Galun, S. Alpert, R. Basri, and G. Shakhnarovich. Viewpoint-aware object detection and pose estimation. In *ICCV*, pages 1275–1282. IEEE, 2011.

[13] C. Gu and X. Ren. Discriminative mixture-of-templates for viewpoint classification. *ECCV*, pages 408–421, 2010.

[14] B. Horn. *Robot vision*. The MIT Press, 1986.

[15] S. Ioffe and D. Forsyth. Mixtures of trees for object recognition. In *CVPR*, 2001.

[16] T. Joachims, T. Finley, and C. Yu. Cutting plane training of structural SVMs. *Machine Learning*, 2009.

[17] M. Jones and P. Viola. Fast multi-view face detection. In *CVPR 2003*.

[18] Y. Li, L. Gu, and T. Kanade. A robust shape model for multi-view car alignment. In *CVPR*, 2009.

[19] R. Lopez-Sastre, T. Tuytelaars, and S. Savarese. Deformable part models revisited: A performance evaluation for object category pose estimation. In *Computer Vision Workshops (ICCV Workshops)*, 2011.

[20] M. Meila and M. Jordan. Learning with mixtures of trees. *JMLR*, 1:1–48, 2001.

[21] P. Ott and M. Everingham. Shared parts for deformable part-based models. In *CVPR*, 2011.

[22] B. Pepik, M. Stark, P. Gehler, and B. Scheile. Teaching geometry to deformable part models. In *CVPR*, 2012.

[23] S. Savarese and L. Fei-Fei. 3d generic object categorization, localization and pose estimation. In *ICCV*, pages 1–8. IEEE, 2007.

[24] H. Schneiderman and T. Kanade. A statistical method for 3d object detection applied to faces and cars. In *CVPR*, volume 1, pages 746–751. IEEE, 2000.

[25] M. Sun, H. Su, S. Savarese, and L. Fei-Fei. A multi-view probabilistic model for 3d object classes. In *CVPR*, pages 1247–1254. IEEE, 2009.

[26] A. Thomas, V. Ferrar, B. Leibe, T. Tuytelaars, B. Schiel, and L. Van Gool. Towards multi-view object class detection. In *CVPR*, volume 2, pages 1589–1596. IEEE, 2006.

[27] A. Torralba, K. Murphy, and W. Freeman. Sharing visual features for multiclass and multiview object detection. *PAMI*, 29(5):854–869, 2007.

[28] L. Torresani, A. Hertzmann, and C. Bregler. Learning non-rigid 3d shape from 2d motion. *Advances in Neural Information Processing Systems*, 16, 2003.

[29] L. Torresani, D. Yang, E. Alexander, and C. Bregler. Tracking and modeling non-rigid objects with rank constraints. In *CVPR*, volume 1, pages I–493. IEEE, 2001.

[30] P. Viola and M. Jones. Rapid object detection using a boosted cascade of simple features. In *CVPR*, volume 1, pages I–511. IEEE, 2001.

[31] Y. Yang and D. Ramanan. Articulated pose estimation with flexible mixtures-of-parts. In *CVPR*, 2011.

[32] L. Zhu, Y. Chen, A. Torralba, W. Freeman, and A. Yuille. Part and appearance sharing: Recursive compositional models for multi-view multi-object detection. *Pattern Recognition*, 2010.

[33] X. Zhu and D. Ramanan. Face detection, pose estimation, and landmark localization in the wild. In *CVPR*, 2012.

[34] M. Zia, M. Stark, B. Schiele, and K. Schindler. Revisiting 3d geometric models for accurate object shape and pose. In *ICCV Workshops*, pages 569–576. IEEE, 2011.

